# Posterior Consistency of the Silverman $g$-prior in Bayesian Model Choice

**Zhihua Zhang**
School of Computer Science & Technology
Zhejiang University, Hangzhou, China

**Michael I. Jordan**
Departments of EECS and Statistics
University of California, Berkeley, CA, USA

**Dit-Yan Yeung**
Department of Computer Science & Engineering
HKUST, Hong Kong, China

## Abstract

Kernel supervised learning methods can be unified by utilizing the tools from regularization theory. The duality between *regularization* and *prior* leads to interpreting regularization methods in terms of maximum *a posteriori* estimation and has motivated Bayesian interpretations of kernel methods. In this paper we pursue a Bayesian interpretation of sparsity in the kernel setting by making use of a mixture of a point-mass distribution and prior that we refer to as "Silverman's $g$-prior." We provide a theoretical analysis of the posterior consistency of a Bayesian model choice procedure based on this prior. We also establish the asymptotic relationship between this procedure and the Bayesian information criterion.

## 1 Introduction

We address a supervised learning problem over a set of training data $\{\mathbf{x}_i, y_i\}_{i=1}^n$ where $\mathbf{x}_i \in \mathcal{X} \subset \mathbb{R}^p$ is a $p$-dimensional input vector and $y_i$ is a univariate response. Using the theory of reproducing kernels, we seek to find a predictive function $f(\mathbf{x})$ from the training data.

Suppose $f = u + h \in (\{1\} + \mathcal{H}_K)$ where $\mathcal{H}_K$ is a reproducing kernel Hilbert space (RKHS). The estimation of $f(\mathbf{x})$ is then formulated as a regularization problem of the form

$$\min_{f \in \mathcal{H}_K} \left\{ \frac{1}{n} \sum_{i=1}^n L(y_i, f(\mathbf{x}_i)) + \frac{g}{2} \|h\|_{\mathcal{H}_K}^2 \right\}, \tag{1}$$

where $L(y, f(\mathbf{x}))$ is a loss function, $\|h\|_{\mathcal{H}_K}^2$ is the RKHS norm and $g > 0$ is the regularization parameter. By the representer theorem [7], the solution for (1) is of the form

$$f(\mathbf{x}) = u + \sum_{j=1}^n \beta_j K(\mathbf{x}, \mathbf{x}_j), \tag{2}$$

where $K(\cdot, \cdot)$ is the kernel function. Noticing that $\|h\|_{\mathcal{H}_K}^2 = \sum_{i,j=1}^n K(\mathbf{x}_i, \mathbf{x}_j)\beta_i\beta_j$ and substituting (2) into (1), we obtain the minimization problem with respect to (w.r.t.) the $\beta_i$ as

$$\min_{u, \boldsymbol{\beta}} \left\{ \frac{1}{n} \sum_{i=1}^n L(y_i, f(\mathbf{x}_i)) + \frac{g}{2} \boldsymbol{\beta}' \mathbf{K} \boldsymbol{\beta} \right\}, \tag{3}$$

where $\mathbf{K} = [K(\mathbf{x}_i, \mathbf{x}_j)]$ is the $n \times n$ kernel matrix and $\boldsymbol{\beta} = (\beta_1, \ldots, \beta_n)'$ is the vector of regression coefficients.

From the Bayesian standpoint, the role of the regularization term $\frac{g}{2}\boldsymbol{\beta}'\mathbf{K}\boldsymbol{\beta}$ can be captured by assigning a design-dependent prior $N_n(\mathbf{0}, g^{-1}\mathbf{K}^{-1})$ to the regression vector $\boldsymbol{\beta}$. The prior $N_n\left(\mathbf{0},\ \mathbf{K}^{-1}\right)$ for $\boldsymbol{\beta}$ was first proposed by [5] in his Bayesian formulation of spline smoothing. Here we refer to the prior $\boldsymbol{\beta} \sim N_n\left(\mathbf{0},\ g^{-1}\mathbf{K}^{-1}\right)$ as the *Silverman g-prior* by analogy to the Zellner g-prior [9]. When $\mathbf{K}$ is singular, by analogy to *generalized singular g-prior* (*gsg*-prior) [8], we call $N_n\left(\mathbf{0},\ g^{-1}\mathbf{K}^{-1}\right)$ a *generalized Silverman g-prior*.

Given the high dimensionality generally associated with RKHS methods, sparseness has emerged as a significant theme, particularly when computational concerns are taken into account. For example, the number of support vectors in support vector machine (SVM) is equal to the number of nonzero components of $\boldsymbol{\beta}$. That is, if $\beta_j = 0$, the $j$th input vector is excluded from the basis expansion in (2); otherwise the $j$th input vector is a support vector. We are thus interested in a prior for $\boldsymbol{\beta}$ which allows some components of $\boldsymbol{\beta}$ to be zero. To specify such a prior we first introduce an indicator vector $\boldsymbol{\gamma} = (\gamma_1, \ldots, \gamma_n)'$ such that $\gamma_j = 1$ if $\mathbf{x}_j$ is a support vector and $\gamma_j = 0$ if it is not. Let $n_\gamma = \sum_{j=1}^n \gamma_j$ be the number of support vectors, let $\mathbf{K}_\gamma$ be the $n \times n_\gamma$ submatrix of $\mathbf{K}$ consisting of those columns of $\mathbf{K}$ for which $\gamma_j = 1$, and let $\boldsymbol{\beta}_\gamma$ be the corresponding subvector of $\boldsymbol{\beta}$. Accordingly, we let $\boldsymbol{\beta}_\gamma \sim N_{n_\gamma}\left(\mathbf{0},\ g^{-1}\mathbf{K}_{\gamma\gamma}^{-1}\right)$ where $\mathbf{K}_{\gamma\gamma}$ is the $n_\gamma \times n_\gamma$ submatrix of $\mathbf{K}_\gamma$ consisting of those rows of $\mathbf{K}_\gamma$ for which $\gamma_j = 1$.

We thus have a Bayesian model choice problem in which a family of models is indexed by an indicator vector $\boldsymbol{\gamma}$. Within the Bayesian framework we can use Bayes factors to choose among these models [3]. In this paper we provide a frequentist theoretical analysis of this Bayesian procedure. In particular, motivated by the work of [1] on the consistency of the Zellner g-prior, we investigate the consistency for model choice of the Silverman g-prior for sparse kernel-based regression.

## 2  Main Results

Our analysis is based on the following regression model $\mathcal{M}_\gamma$:

$$\mathbf{y} = u\mathbf{1}_n + \mathbf{K}_\gamma\boldsymbol{\beta}_\gamma + \boldsymbol{\epsilon} \tag{4}$$

$$\boldsymbol{\epsilon} \sim N_n(\mathbf{0},\ \sigma^2\mathbf{I}_n), \quad \boldsymbol{\beta}_\gamma|\sigma \sim N_{n_\gamma}\left(\mathbf{0},\ \sigma^2(g_\gamma\mathbf{K}_{\gamma\gamma})^{-1}\right),$$

where $\mathbf{y} = (y_1, \ldots, y_n)'$. Here and later, $\mathbf{1}_m$ denotes the $m \times 1$ vector of ones and $\mathbf{I}_m$ denotes the $m \times m$ identity matrix. We compare each model $\mathcal{M}_\gamma$ with the null model $\mathcal{M}_0$, formulating the model choice problem via the hypotheses $H_0 : \boldsymbol{\beta} = \mathbf{0}$ and $H_\gamma : \boldsymbol{\beta}_\gamma \in \mathbb{R}^{n_\gamma}$.

Throughout this paper, for any $n_\gamma$, it is always assumed to take a finite value even though $n \to \infty$. Let $\widetilde{\mathbf{K}}_\gamma = [\mathbf{1}_n, \mathbf{K}_\gamma]$. The following condition is also assumed:

$$\begin{array}{l} \text{For a fixed } n_\gamma < n,\ \frac{1}{n}\widetilde{\mathbf{K}}_\gamma'\widetilde{\mathbf{K}}_\gamma \text{ is positive definite and} \\ \text{converges to a positive definite matrix as } n \to \infty. \end{array} \tag{5}$$

Suppose that the sample $\mathbf{y}$ is generated by model $\mathcal{M}_\nu$ with parameter values $u$, $\boldsymbol{\beta}_\nu$ and $\sigma$. We formalize the problem of consistency for model choice as follows [1]:

$$\operatorname*{plim}_{n\to\infty} p(\mathcal{M}_\nu|\mathbf{y}) = 1 \text{ and } \operatorname*{plim}_{n\to\infty} p(\mathcal{M}_\gamma|\mathbf{y}) = 0 \text{ for all } \mathcal{M}_\gamma \neq \mathcal{M}_\nu, \tag{6}$$

where "plim" denotes convergence in probability and the limit is taken w.r.t. the sampling distribution under the true model $\mathcal{M}_\nu$.

### 2.1  A Noninformative Prior for $(u, \sigma^2)$

We first consider the case when $(u, \sigma^2)$ is assigned the following noninformative prior:

$$(u, \sigma^2) \propto 1/\sigma^2. \tag{7}$$

Moreover, we assume $\mathbf{1}_n'\mathbf{K} = \mathbf{0}$. In this case, we have $\mathbf{1}_n'\mathbf{K}_\gamma = \mathbf{0}$ so that the intercept $u$ may be regarded as a common parameter for both $\mathcal{M}_\gamma$ and $\mathcal{M}_0$.

After some calculations the marginal likelihood is found to be

$$p(\mathbf{y}|\mathcal{M}_\gamma) = \frac{\Gamma(\frac{n-1}{2})}{\pi^{\frac{n-1}{2}}\sqrt{n}}\|\mathbf{y} - \bar{y}\mathbf{1}_n\|^{-n+1}|\mathbf{Q}_\gamma|^{-\frac{1}{2}}(1 - F_\gamma^2)^{-\frac{n-1}{2}}, \tag{8}$$

where $\bar{y} = \frac{1}{n}\sum_{i=1}^{n} y_i$, $\mathbf{Q}_\gamma = \mathbf{I}_n + g_\gamma^{-1}\mathbf{K}_\gamma\mathbf{K}_{\gamma\gamma}^{-1}\mathbf{K}_\gamma'$ and

$$F_\gamma^2 = \frac{\mathbf{y}'\mathbf{K}_\gamma(g_\gamma\mathbf{K}_{\gamma\gamma} + \mathbf{K}_\gamma'\mathbf{K}_\gamma)^{-1}\mathbf{K}_\gamma'\mathbf{y}}{\|\mathbf{y} - \bar{y}\mathbf{1}_n\|^2}.$$

Let $\mathrm{RSS}_\gamma = (1 - R_\gamma^2)\|\mathbf{y} - \bar{y}\mathbf{1}_n\|^2$ be the residual sum of squares. Here,

$$R_\gamma^2 = \frac{(\mathbf{y} - \bar{y}\mathbf{1}_n)'\mathbf{K}_\gamma(\mathbf{K}_\gamma'\mathbf{K}_\gamma)^{-1}\mathbf{K}_\gamma'(\mathbf{y} - \bar{y}\mathbf{1}_n)}{\|\mathbf{y} - \bar{y}\mathbf{1}_n\|^2} = \frac{\mathbf{y}'\mathbf{K}_\gamma(\mathbf{K}_\gamma'\mathbf{K}_\gamma)^{-1}\mathbf{K}_\gamma'\mathbf{y}}{\|\mathbf{y} - \bar{y}\mathbf{1}_n\|^2}.$$

It is easily proven that for fixed $n$, $\mathrm{plim}_{g_\gamma\to 0} F_\gamma^2 = R_\gamma^2$ and $\mathrm{plim}_{g_\gamma\to 0}(1 - F_\gamma^2)\|\mathbf{y} - \bar{y}\mathbf{1}_n\|^2 = \mathrm{RSS}_\gamma$, and $\mathrm{RSS}_\gamma = \mathbf{y}'(\mathbf{I}_n - \widetilde{\mathbf{H}}_\gamma)\mathbf{y}$ where $\widetilde{\mathbf{H}}_\gamma = \widetilde{\mathbf{K}}_\gamma(\widetilde{\mathbf{K}}_\gamma'\widetilde{\mathbf{K}}_\gamma)^{-1}\widetilde{\mathbf{K}}_\gamma'$. As a special case of (8), it is also immediate to obtain the marginal distribution of the null model as

$$p(\mathbf{y}|\mathcal{M}_0) = \frac{\Gamma(\frac{n-1}{2})}{\pi^{\frac{n-1}{2}}\sqrt{n}}\|\mathbf{y} - \bar{y}\mathbf{1}_n\|^{-n+1}.$$

Then the Bayes factor for $\mathcal{M}_\gamma$ versus $\mathcal{M}_0$ is

$$\mathrm{BF}_{\gamma 0} = |\mathbf{Q}_\gamma|^{-\frac{1}{2}}(1 - F_\gamma^2)^{-\frac{n-1}{2}}.$$

In the limiting case when $g_\gamma \to 0$ and both $n$ and $n_\gamma$ are fixed, $\mathrm{BF}_{\gamma 0}$ tends to 0. This implies that a large spread of the prior forces the Bayes factor to favor the null model. Thus, as in the case of the Zellner $g$-prior [4], Bartlett's paradox arises for the Silverman $g$-prior.

The Bayes factor for $\mathcal{M}_\gamma$ versus $\mathcal{M}_\kappa$ is given by

$$\mathrm{BF}_{\gamma\kappa} = \frac{\mathrm{BF}_{\gamma 0}}{\mathrm{BF}_{\kappa 0}} = \frac{|\mathbf{Q}_\gamma|^{-\frac{1}{2}}}{|\mathbf{Q}_\kappa|^{-\frac{1}{2}}}\frac{(1 - F_\gamma^2)^{-\frac{n-1}{2}}}{(1 - F_\kappa^2)^{-\frac{n-1}{2}}}. \tag{9}$$

Based on the Bayes factor, we now explore the consistency of the Silverman $g$-prior. Suppose that the sample $\mathbf{y}$ is generated by model $\mathcal{M}_\nu$ with parameter values $u$, $\boldsymbol{\beta}_\nu$ and $\sigma^2$. Then the consistency property (6) is equivalent to

$$\mathrm{plim}_{n\to\infty}\ \mathrm{BF}_{\gamma\nu} = 0, \quad \text{for all } \mathcal{M}_\gamma \neq \mathcal{M}_\nu.$$

Assume that under any model $\mathcal{M}_\gamma$ that does not contain $\mathcal{M}_\nu$, i.e, $\mathcal{M}_\gamma \not\supseteq \mathcal{M}_\nu$,

$$\lim_{n\to\infty}\frac{\widetilde{\boldsymbol{\beta}}_\gamma'\widetilde{\mathbf{K}}_\nu'(\mathbf{I}_n - \widetilde{\mathbf{H}}_\gamma)\widetilde{\mathbf{K}}_\nu\widetilde{\boldsymbol{\beta}}_\gamma}{n} = c_\gamma \in (0, \infty), \tag{10}$$

where $\widetilde{\boldsymbol{\beta}}_\gamma' = (u, \boldsymbol{\beta}_\gamma')$. Note that $\mathbf{I}_n - \widetilde{\mathbf{H}}_\gamma$ is a symmetric idempotent matrix which projects onto the subspace of $\mathbb{R}^n$ orthogonal to the span of $\widetilde{\mathbf{K}}_\gamma$. Given that $(\mathbf{I}_n - \widetilde{\mathbf{H}}_\gamma)\mathbf{1}_n = \mathbf{0}$ and $\mathbf{1}_n'\mathbf{K}_\nu = \mathbf{0}$, condition (10) reduces to

$$\lim_{n\to\infty}\frac{\boldsymbol{\beta}_\nu'\mathbf{K}_\nu'(\mathbf{I}_n - \mathbf{H}_\gamma)\mathbf{K}_\nu\boldsymbol{\beta}_\nu}{n} = c_\gamma \in (0, \infty),$$

where $\mathbf{H}_\gamma = \mathbf{K}_\gamma(\mathbf{K}_\gamma'\mathbf{K}_\gamma)^{-1}\mathbf{K}_\gamma'$. We now have the following theorem whose proof is given in Sec. 3.

**Theorem 1** *Consider the regression model (4) with the noninformative prior for $(u, \sigma^2)$ in (7). Assume that conditions (5) and (10) are satisfied and assume that $g_\gamma$ can be written in the form*

$$g_\gamma = \frac{w_1(n_\gamma)}{w_2(n)} \quad \text{with} \quad \lim_{n\to\infty} w_2(n) = \infty \quad \text{and} \quad \lim_{n\to\infty}\frac{w_2'(n)}{w_2(n)} = 0 \tag{11}$$

*for particular choices of functions $w_1$ and $w_2$, where $w_2$ is differentiable and $w_2'(n)$ is the first derivative w.r.t. $n$. When the true model $\mathcal{M}_\nu$ is not the null model, i.e., $\mathcal{M}_\nu \neq \mathcal{M}_0$, the posterior probabilities are consistent for model choice.*

Theorem 1 can provide an empirical methodology for setting $g$. For example, it is clear that $g = 1/n$ where $w_1(n_\gamma) = 1$ and $w_2(n) = n$ satisfies condition (11).

It is interesting to consider the (asymptotic) relationship between the Bayes factor and Bayesian information (or Schwartz) criterion (BIC) in our setting. Given two models $\mathcal{M}_\gamma$ and $\mathcal{M}_\kappa$, the difference between the BICs of these two models is given by

$$S_{\gamma\kappa} = \frac{n}{2} \ln \frac{\text{RSS}_\kappa}{\text{RSS}_\gamma} + \frac{n_\kappa - n_\gamma}{2} \ln(n).$$

We thus obtain the following asymptotic relationship (the proof is given in Sec. 3):

**Theorem 2** *Under the regression model and the conditions in Theorem 1, we have*

$$\plim_{n \to \infty} \frac{\ln \text{BF}_{\gamma\nu}}{S_{\gamma\nu} + \frac{n_\nu - n_\gamma}{2} \ln w_2(n)} = 1.$$

*Furthermore, if $\mathcal{M}_\nu$ is not nested within $\mathcal{M}_\gamma$, then $\plim_{n \to \infty} \frac{\ln \text{BF}_{\gamma\nu}}{S_{\gamma\nu}} = 1$. Here the probability limits are taken w.r.t. the model $\mathcal{M}_\nu$.*

## 2.2 A Natural Conjugate Prior for $(u, \sigma^2)$

In this section, we analyze consistency for model choice under a different prior for $(u, \sigma^2)$, namely the standard conjugate prior:

$$p(u, \sigma^2) = N(u|0,\ \sigma^2\eta^{-1})\, Ga(\sigma^{-2}|a_\sigma/2,\ b_\sigma/2) \tag{12}$$

where $Ga(u|a, b)$ is the Gamma distribution:

$$p(u) = \frac{b^a}{\Gamma(a)} u^{a-1} \exp(-bu),\ a > 0, b > 0.$$

We further assume that $u$ and $\boldsymbol{\beta}_\gamma$ are independent. Then

$$\widetilde{\boldsymbol{\beta}}_\gamma \sim N_{n_\gamma+1}(\mathbf{0},\ \sigma^2 \boldsymbol{\Sigma}_\gamma^{-1}) \ \text{ with } \ \boldsymbol{\Sigma}_\gamma = \begin{bmatrix} \eta & \mathbf{0} \\ \mathbf{0} & g_\gamma \mathbf{K}_{\gamma\gamma} \end{bmatrix}. \tag{13}$$

The marginal likelihood of model $\mathcal{M}_\gamma$ is thus

$$p(\mathbf{y}|\mathcal{M}_\gamma) = \frac{b_\sigma^{a_\sigma/2} \Gamma(\frac{n+a_\sigma}{2})}{\pi^{n/2} \Gamma(\frac{a_\sigma}{2})} |\mathbf{M}_\gamma|^{-\frac{1}{2}} \left[ b_\sigma + \mathbf{y}'\mathbf{M}_\gamma^{-1}\mathbf{y} \right]^{-\frac{a_\sigma+n}{2}}, \tag{14}$$

where $\mathbf{M}_\gamma = \mathbf{I}_n + \widetilde{\mathbf{K}}_\gamma \boldsymbol{\Sigma}_\gamma^{-1} \widetilde{\mathbf{K}}_\gamma'$. The Bayes factor for $\mathcal{M}_\gamma$ versus $\mathcal{M}_\kappa$ is given by

$$\text{BF}_{\gamma\kappa} = \left[ \frac{|\mathbf{M}_\kappa|}{|\mathbf{M}_\gamma|} \right]^{\frac{1}{2}} \left[ \frac{b_\sigma + \mathbf{y}'\mathbf{M}_\kappa^{-1}\mathbf{y}}{b_\sigma + \mathbf{y}'\mathbf{M}_\gamma^{-1}\mathbf{y}} \right]^{\frac{a_\sigma+n}{2}}.$$

Because $\mathbf{M}_\gamma^{-1} = \mathbf{I}_n - \widetilde{\mathbf{K}}_\gamma \boldsymbol{\Theta}_\gamma^{-1} \widetilde{\mathbf{K}}_\gamma'$ and $|\mathbf{M}_\gamma| = |\boldsymbol{\Theta}_\gamma| |\boldsymbol{\Sigma}_\gamma|^{-1} = \eta^{-1} g_\gamma^{-n_\gamma} |\mathbf{K}_{\gamma\gamma}|^{-1} |\boldsymbol{\Theta}_\gamma|$ where $\boldsymbol{\Theta}_\gamma = \widetilde{\mathbf{K}}_\gamma' \widetilde{\mathbf{K}}_\gamma + \boldsymbol{\Sigma}_\gamma$, we have

$$\text{BF}_{\gamma\kappa} = \frac{g_\gamma^{n_\gamma/2}}{g_\kappa^{n_\kappa/2}} \left[ \frac{|\mathbf{K}_{\gamma\gamma}||\boldsymbol{\Theta}_\kappa|}{|\mathbf{K}_{\kappa\kappa}||\boldsymbol{\Theta}_\gamma|} \right]^{\frac{1}{2}} \left[ \frac{b_\sigma + \mathbf{y}'\big(\mathbf{I}_n - \widetilde{\mathbf{K}}_\kappa \boldsymbol{\Theta}_\kappa^{-1} \widetilde{\mathbf{K}}_\kappa'\big)\mathbf{y}}{b_\sigma + \mathbf{y}'\big(\mathbf{I}_n - \widetilde{\mathbf{K}}_\gamma \boldsymbol{\Theta}_\gamma^{-1} \widetilde{\mathbf{K}}_\gamma'\big)\mathbf{y}} \right]^{\frac{a_\sigma+n}{2}}.$$

**Theorem 3** *Consider the regression model (4) with the conjugate prior for $(u, \sigma^2)$ in (12). Assume that conditions (5) and (10) are satisfied and that $g_\gamma$ takes the form in (11) with $w_1(n_\gamma)$ being a decreasing function. When the true model $\mathcal{M}_\nu$ is not the null model, i.e., $\mathcal{M}_\nu \neq \mathcal{M}_0$, the posterior probabilities are consistent for model choice.*

Note the difference between Theorem 1 and Theorem 3: in the latter theorem $w_1(n_\gamma)$ is required to be a decreasing function of $n_\gamma$. Thanks to the fact that $g_\gamma = w_1(n_\gamma)/w_2(n)$, such a condition is equivalent to assuming that $g_\gamma$ is a decreasing function of $n_\gamma$. Again, $g_\gamma = 1/n$ satisfies these conditions. Similarly with Theorem 2, we also have

**Theorem 4** *Under the regression model and the conditions in Theorem 3, we have*

$$\operatorname*{plim}_{n\to\infty} \frac{\ln \mathrm{BF}_{\gamma\nu}}{\mathrm{S}_{\gamma\nu} + \frac{n_\nu - n_\gamma}{2} \ln w_2(n)} = 1.$$

*Furthermore, if $\mathcal{M}_\nu$ is not nested within $\mathcal{M}_\gamma$, then $\operatorname{plim}_{n\to\infty} \frac{\ln \mathrm{BF}_{\gamma\nu}}{\mathrm{S}_{\gamma\nu}} = 1$. Here the probability limits are taken w.r.t. the model $\mathcal{M}_\nu$.*

## 3 Proofs

In order to prove these theorems, we first give the following lemmas.

**Lemma 1** *Let $\mathbf{A} = \begin{bmatrix} \mathbf{A}_{11} & \mathbf{A}_{12} \\ \mathbf{A}_{21} & \mathbf{A}_{22} \end{bmatrix}$ be symmetric and positive definite, and let $\mathbf{B} = \begin{bmatrix} \mathbf{A}_{11}^{-1} & \mathbf{0} \\ \mathbf{0} & \mathbf{0} \end{bmatrix}$ have the same size as $\mathbf{A}$. Then $\mathbf{A}^{-1} - \mathbf{B}$ is positive semidefinite.*

**Proof** The proof follows readily once we express $\mathbf{A}^{-1}$ and $\mathbf{B}$ as

$$\mathbf{A}^{-1} = \begin{bmatrix} \mathbf{I} & -\mathbf{A}_{11}^{-1}\mathbf{A}_{12} \\ \mathbf{0} & \mathbf{I} \end{bmatrix} \begin{bmatrix} \mathbf{A}_{11}^{-1} & \mathbf{0} \\ \mathbf{0} & \mathbf{A}_{22\cdot1}^{-1} \end{bmatrix} \begin{bmatrix} \mathbf{I} & \mathbf{0} \\ -\mathbf{A}_{21}\mathbf{A}_{11}^{-1} & \mathbf{I} \end{bmatrix},$$

$$\mathbf{B} = \begin{bmatrix} \mathbf{I} & -\mathbf{A}_{11}^{-1}\mathbf{A}_{12} \\ \mathbf{0} & \mathbf{I} \end{bmatrix} \begin{bmatrix} \mathbf{A}_{11}^{-1} & \mathbf{0} \\ \mathbf{0} & \mathbf{0} \end{bmatrix} \begin{bmatrix} \mathbf{I} & \mathbf{0} \\ -\mathbf{A}_{21}\mathbf{A}_{11}^{-1} & \mathbf{I} \end{bmatrix},$$

where $\mathbf{A}_{22\cdot1} = \mathbf{A}_{22} - \mathbf{A}_{21}\mathbf{A}_{11}^{-1}\mathbf{A}_{12}$ is also positive definite. ∎

The following two lemmas were presented by [1].

**Lemma 2** *Under the sampling model $\mathcal{M}_\nu$: (i) if $\mathcal{M}_\nu$ is nested within or equal to a model $\mathcal{M}_\gamma$, i.e., $\mathcal{M}_\nu \subseteq \mathcal{M}_\gamma$, then*

$$\operatorname*{plim}_{n\to\infty} \frac{\mathrm{RSS}_\gamma}{n} = \sigma^2$$

*and (ii) for any model $\mathcal{M}_\gamma$ that does not contain $\mathcal{M}_\nu$, if (10) satisfies, then*

$$\operatorname*{plim}_{n\to\infty} \frac{\mathrm{RSS}_\gamma}{n} = \sigma^2 + c_\gamma.$$

**Lemma 3** *Under the sampling model $\mathcal{M}_\nu$, if $\mathcal{M}_\nu$ is nested within a model $\mathcal{M}_\gamma$, i.e., $\mathcal{M}_\nu \subset \mathcal{M}_\gamma$, then $n \ln\left(\frac{\mathrm{RSS}_\nu}{\mathrm{RSS}_\gamma}\right) \xrightarrow{d} \chi^2_{n_\gamma - n_\nu}$ as $n \to \infty$ where $\xrightarrow{d}$ denotes convergence in distribution.*

**Lemma 4** *Under the regression model (4), if $\lim_{n\to\infty} g_\gamma(n) = 0$ and condition (5) is satisfied, then*

$$\operatorname*{plim}_{n\to\infty} (1 - F_\gamma^2)\|\mathbf{y} - \bar{y}\mathbf{1}_n\|^2 - \mathrm{RSS}_\gamma = 0.$$

**Proof** It is easy to compute

$$\frac{(1 - F_\gamma^2)\|\mathbf{y} - \bar{y}\mathbf{1}_n\|^2 - \mathrm{RSS}_\gamma}{\sigma^2} = \frac{\mathbf{y}'\mathbf{K}_\gamma[(\mathbf{K}'_\gamma\mathbf{K}_\gamma)^{-1} - (\mathbf{K}'_\gamma\mathbf{K}_\gamma + g_\gamma(n)\mathbf{K}_{\gamma\gamma})^{-1}]\mathbf{K}'_\gamma\mathbf{y}}{\sigma^2}.$$

Since both $\mathbf{K}'_\gamma\mathbf{K}_\gamma/n$ and $\mathbf{K}_{\gamma\gamma}$ are positive definite, there exists an $n_\gamma \times n_\gamma$ nonsingular matrix $\mathbf{A}_n$ and an $n_\gamma \times n_\gamma$ positive diagonal matrix $\mathbf{\Lambda}_{n_\gamma}$ such that $\mathbf{K}'_\gamma\mathbf{K}_\gamma/n = \mathbf{A}'_n\mathbf{\Lambda}_{n_\gamma}\mathbf{A}_n$ and $\mathbf{K}_{\gamma\gamma} = \mathbf{A}'_n\mathbf{A}_n$. Letting $\mathbf{z} = \sigma^{-1}(n\mathbf{\Lambda}_{n_\gamma})^{-1/2}(\mathbf{A}'_n)^{-1}\mathbf{K}'_\gamma\mathbf{y}$, we have

$$\mathbf{z} \sim N_{n_\gamma}(\sigma^{-1}(n\mathbf{\Lambda}_{n_\gamma})^{1/2}\mathbf{A}_n\boldsymbol{\beta}, \ \mathbf{I}_{n_\gamma})$$

and

$$f(\mathbf{z}) \triangleq \frac{(1 - F_\gamma^2)\|\mathbf{y} - \bar{y}\mathbf{1}_n\|^2 - \mathrm{RSS}_\gamma}{\sigma^2} = \mathbf{z}'\mathbf{z} - \mathbf{z}'n\mathbf{\Lambda}_{n_\gamma}\left[n\mathbf{\Lambda}_{n_\gamma} + g_\gamma(n)\mathbf{I}_{n_\gamma}\right]^{-1}\mathbf{z}$$

$$= \sum_{j=1}^{n_\gamma} \frac{g_\gamma(n)}{n\lambda_j(n) + g_\gamma(n)} z_j^2.$$

Note that $z_j^2$ follows a noncentral chi-square distribution, $\chi^2(1, v_j)$, with $v_j =$ $n\lambda_j(n)(\mathbf{a}_j(n)'\boldsymbol{\beta})^2/\sigma^2$ where $\lambda_j(n) > 0$ is the $j$th diagonal element of $\boldsymbol{\Lambda}_{n_\gamma}$ and $\mathbf{a}_j(n)$ is the $j$th column of $\mathbf{A}_n$. We thus have $\mathrm{E}(z_j^2) = 1 + v_j$ and $\mathrm{Var}(z_j^2) = 2(1 + 2v_j)$. It follows from condition (5) that

$$\lim_{n\to\infty} \mathbf{K}_\gamma'\mathbf{K}_\gamma/n = \lim_{n\to\infty} \mathbf{A}_n'\boldsymbol{\Lambda}_{n_\gamma}\mathbf{A}_n = \mathbf{A}'\boldsymbol{\Lambda}_\gamma\mathbf{A},$$

where $\mathbf{A}$ is nonsingular and $\boldsymbol{\Lambda}_\gamma$ is a diagonal matrix with positive diagonal elements, and both are independent of $n$. Hence,

$$\lim_{n\to\infty} \mathrm{E}\Big(\frac{g_\gamma(n)}{n\lambda_j(n) + g_\gamma(n)}z_j^2\Big) = 0 \ \text{ and } \ \lim_{n\to\infty} \mathrm{Var}\Big(\frac{g_\gamma(n)}{n\lambda_j(n) + g_\gamma(n)}z_j^2\Big) = 0.$$

We thus have $\mathrm{plim}_{n\to\infty} f(\mathbf{z}) = 0$. The proof is completed. ∎

**Lemma 5** *Assume that $\mathcal{M}_\kappa$ is nested within $\mathcal{M}_\gamma$ and $g_\gamma$ is a decreasing function of $n_\gamma$. Then*

$$\mathbf{y}'(\mathbf{I}_n - \widetilde{\mathbf{K}}_\kappa\boldsymbol{\Theta}_\kappa^{-1}\widetilde{\mathbf{K}}_\kappa')\mathbf{y} \geq \mathbf{y}'(\mathbf{I}_n - \widetilde{\mathbf{K}}_\gamma\boldsymbol{\Theta}_\gamma^{-1}\widetilde{\mathbf{K}}_\gamma')\mathbf{y}.$$

**Proof** Since $\mathcal{M}_\kappa$ is nested within $\mathcal{M}_\gamma$, we express $\widetilde{\mathbf{K}}_\gamma = [\widetilde{\mathbf{K}}_\kappa, \mathbf{K}_2]$ without loss of generality. We now write $\boldsymbol{\Sigma}_\gamma = \begin{bmatrix} \boldsymbol{\Sigma}_\gamma^{11} & \boldsymbol{\Sigma}_\gamma^{12} \\ \boldsymbol{\Sigma}_\gamma^{21} & \boldsymbol{\Sigma}_\gamma^{22} \end{bmatrix}$ where $\boldsymbol{\Sigma}_\gamma^{11}$ is of size $n_\kappa \times n_\kappa$. Hence, we have

$$\boldsymbol{\Theta}_\gamma^{-1} = \begin{bmatrix} \widetilde{\mathbf{K}}_\kappa'\widetilde{\mathbf{K}}_\kappa + \boldsymbol{\Sigma}_\gamma^{11} & \widetilde{\mathbf{K}}_\kappa'\mathbf{K}_2 + \boldsymbol{\Sigma}_\gamma^{12} \\ \mathbf{K}_2'\widetilde{\mathbf{K}}_\kappa + \boldsymbol{\Sigma}_\gamma^{21} & \mathbf{K}_2'\mathbf{K}_2 + \boldsymbol{\Sigma}_\gamma^{22} \end{bmatrix}^{-1}.$$

Because $0 < g_\gamma \leq g_\kappa$, $\widetilde{\mathbf{K}}_\kappa'\widetilde{\mathbf{K}}_\kappa + \boldsymbol{\Sigma}_\kappa - (\widetilde{\mathbf{K}}_\kappa'\widetilde{\mathbf{K}}_\kappa + \boldsymbol{\Sigma}_\gamma^{11}) = \begin{bmatrix} \mathbf{0} & \mathbf{0} \\ \mathbf{0} & (g_\kappa - g_\gamma)\mathbf{K}_{\kappa\kappa} \end{bmatrix}$ is positive semidefinite. Consequently, $(\widetilde{\mathbf{K}}_\kappa'\widetilde{\mathbf{K}}_\kappa + \boldsymbol{\Sigma}_\gamma^{11})^{-1} - (\widetilde{\mathbf{K}}_\kappa'\widetilde{\mathbf{K}}_\kappa + \boldsymbol{\Sigma}_\kappa)^{-1}$ is positive semidefinite. It follows from Lemma 1 that $\boldsymbol{\Theta}_\gamma^{-1} - \begin{bmatrix} (\widetilde{\mathbf{K}}_\kappa'\widetilde{\mathbf{K}}_\kappa + \boldsymbol{\Sigma}_\kappa)^{-1} & \mathbf{0} \\ \mathbf{0} & \mathbf{0} \end{bmatrix}$ is also positive semidefinite. We thus have

$$\mathbf{y}'(\mathbf{I}_n - \widetilde{\mathbf{K}}_\kappa\boldsymbol{\Theta}_\kappa^{-1}\widetilde{\mathbf{K}}_\kappa')\mathbf{y} - \mathbf{y}'(\mathbf{I}_n - \widetilde{\mathbf{K}}_\gamma\boldsymbol{\Theta}_\gamma^{-1}\widetilde{\mathbf{K}}_\gamma')\mathbf{y}$$

$$= \mathbf{y}'\widetilde{\mathbf{K}}_\gamma\Bigg(\begin{bmatrix} \widetilde{\mathbf{K}}_\kappa'\widetilde{\mathbf{K}}_\kappa + \boldsymbol{\Sigma}_\gamma^{11} & \widetilde{\mathbf{K}}_\kappa'\mathbf{K}_2 + \boldsymbol{\Sigma}_\gamma^{12} \\ \mathbf{K}_2'\widetilde{\mathbf{K}}_\kappa + \boldsymbol{\Sigma}_\gamma^{21} & \mathbf{K}_2'\mathbf{K}_2 + \boldsymbol{\Sigma}_\gamma^{22} \end{bmatrix}^{-1} - \begin{bmatrix} (\widetilde{\mathbf{K}}_\kappa'\widetilde{\mathbf{K}}_\kappa + \boldsymbol{\Sigma}_\kappa)^{-1} & \mathbf{0} \\ \mathbf{0} & \mathbf{0} \end{bmatrix}\Bigg)\widetilde{\mathbf{K}}_\gamma'\mathbf{y} \geq 0.$$

∎

### 3.1 Proof of Theorem 1

We now prove Theorem 1. Consider that

$$\ln\mathrm{BF}_{\gamma\nu} = \frac{1}{2}\ln\frac{|\mathbf{Q}_\nu|}{|\mathbf{Q}_\gamma|} + \frac{n-1}{2}\ln\frac{(1 - F_\nu^2)}{(1 - F_\gamma^2)}.$$

Because

$$|\mathbf{Q}_\gamma|^{-\frac{1}{2}} = \frac{g_\gamma^{\frac{n_\gamma}{2}}|\mathbf{K}_{\gamma\gamma}|^{1/2}}{|g_\gamma\mathbf{K}_{\gamma\gamma} + \mathbf{K}_\gamma'\mathbf{K}_\gamma|^{1/2}},$$

we have

$$\ln\frac{|\mathbf{Q}_\nu|}{|\mathbf{Q}_\gamma|} = \ln\frac{w_1(n_\gamma)^{n_\gamma}}{w_1(n_\nu)^{n_\nu}} + \ln\frac{|\mathbf{K}_{\gamma\gamma}|}{|\mathbf{K}_{\nu\nu}|} + \ln\frac{\big|\frac{w_1(n_\nu)}{nw_2(n)}\mathbf{K}_{\nu\nu} + \frac{1}{n}\mathbf{K}_\nu'\mathbf{K}_\nu\big|}{\big|\frac{w_1(n_\gamma)}{nw_2(n)}\mathbf{K}_{\gamma\gamma} + \frac{1}{n}\mathbf{K}_\gamma'\mathbf{K}_\gamma\big|} + (n_\nu - n_\gamma)\ln(nw_2(n)).$$

Because

$$\alpha = \lim_{n\to\infty}\ln\frac{\big|\frac{w_1(n_\nu)}{nw_2(n)}\mathbf{K}_{\nu\nu} + \frac{1}{n}\mathbf{K}_\nu'\mathbf{K}_\nu\big|}{\big|\frac{w_1(n_\gamma)}{nw_2(n)}\mathbf{K}_{\gamma\gamma} + \frac{1}{n}\mathbf{K}_\gamma'\mathbf{K}_\gamma\big|} = \lim_{n\to\infty}\ln\frac{\big|\frac{1}{n}\mathbf{K}_\nu'\mathbf{K}_\nu\big|}{\big|\frac{1}{n}\mathbf{K}_\gamma'\mathbf{K}_\gamma\big|} \in (-\infty, \infty),$$

it is easily proven that

$$\lim_{n\to\infty} \frac{1}{2}\ln\frac{|\mathbf{Q}_\nu|}{|\mathbf{Q}_\gamma|} = \left\{ \begin{array}{ll} \infty & n_\gamma < n_\nu \\ -\infty & n_\gamma > n_\nu \\ \text{const} & n_\gamma = n_\nu, \end{array} \right. \tag{15}$$

where const $= \frac{\alpha}{2} + \frac{1}{2}\ln\frac{|\mathbf{K}_{\gamma\gamma}|}{|\mathbf{K}_{\nu\nu}|}$. According to Lemma 4, we also have

$$\operatorname*{plim}_{n\to\infty} \frac{n-1}{2}\ln\frac{(1-F_\nu^2)}{(1-F_\gamma^2)} = \operatorname*{plim}_{n\to\infty} \frac{n-1}{2}\ln\frac{(1-F_\nu^2)\|\mathbf{y}-\bar{y}\mathbf{1}_n\|^2}{(1-F_\gamma^2)\|\mathbf{y}-\bar{y}\mathbf{1}_n\|^2} = \operatorname*{plim}_{n\to\infty} \frac{n-1}{2}\ln\frac{\mathrm{RSS}_\nu}{\mathrm{RSS}_\gamma}.$$

Now consider the following two cases:

(a) $\mathcal{M}_\nu$ is not nested within $\mathcal{M}_\gamma$:
From Lemma 2, we obtain

$$\operatorname*{plim}_{n\to\infty} \ln\frac{\mathrm{RSS}_\nu}{\mathrm{RSS}_\gamma} = \operatorname*{plim}_{n\to\infty} \ln\frac{\mathrm{RSS}_\nu/n}{\mathrm{RSS}_\gamma/n} = \ln\frac{\sigma^2}{\sigma^2+c_\gamma}.$$

Moreover, we have the following limit

$$\lim_{n\to\infty} \frac{n-1}{2}\left[\ln\left(\frac{\sigma^2}{\sigma^2+c_\gamma}\right) + \frac{n_\nu-n_\gamma}{n-1}\ln(nw_2(n))\right] = -\infty$$

due to $\lim_{n\to\infty} \frac{n_\nu-n_\gamma}{n-1}\ln(nw_2(n)) = \lim_{n\to\infty}(n_\nu-n_\gamma)\frac{w_2(n)+nw_2'(n)}{nw_2(n)} = 0$ and $\ln\left(\frac{\sigma^2}{\sigma^2+c_\gamma}\right) < 1$. This implies that $\lim_{n\to\infty}\ln\mathrm{BF}_{\gamma\nu} = -\infty$. Thus we obtain $\lim_{n\to\infty}\mathrm{BF}_{\gamma\nu} = 0$.

(b) $\mathcal{M}_\nu$ is nested within $\mathcal{M}_\gamma$:
We always have $n_\gamma > n_\nu$. By Lemma 3, we have $(n-1)\ln(\mathrm{RSS}_\nu/\mathrm{RSS}_\gamma) \xrightarrow{d} \chi^2_{n_\gamma-n_\nu}$. Hence, $(\mathrm{RSS}_\nu/\mathrm{RSS}_\gamma)^{(n-1)/2} \xrightarrow{d} \exp(\chi^2_{n_\gamma-n_\nu}/2)$. Combining this result with (15) leads to a zero limit for $\mathrm{BF}_{\gamma\nu}$.

### 3.2 Proof of Theorem 2

Using the same notations as those in Theorem 1, we have

$$C_{\gamma\nu} = \frac{\ln\mathrm{BF}_{\gamma\nu}}{S_{\gamma\nu} + \frac{n_\nu-n_\gamma}{2}\ln w_2(n)} = \frac{\frac{n-1}{n}\ln\frac{(1-F_\nu^2)}{(1-F_\gamma^2)} + \frac{n_\nu-n_\gamma}{n}\ln(nw_2(n)) + \frac{2}{n}\mathrm{Const}}{\ln\frac{\mathrm{RSS}_\nu}{\mathrm{RSS}_\gamma} + \frac{n_\nu-n_\gamma}{n}\ln(nw_2(n))}.$$

(a) $\mathcal{M}_\nu$ is not nested within $\mathcal{M}_\gamma$:
From Lemma 4, we obtain

$$\operatorname*{plim}_{n\to\infty} C_{\gamma\nu} = \lim_{n\to\infty} \frac{\ln\frac{\sigma^2}{\sigma^2+c_\gamma} + \frac{n_\nu-n_\gamma}{n}\ln(nw_2(n))}{\ln\frac{\sigma^2}{\sigma^2+c_\gamma} + \frac{n_\nu-n_\gamma}{n}\ln(nw_2(n))} = 1.$$

In this case, we also have

$$\operatorname*{plim}_{n\to\infty} \frac{\ln\mathrm{BF}_{\gamma\nu}}{S_{\gamma\nu}} = \lim_{n\to\infty} \frac{\ln\frac{\sigma^2}{\sigma^2+c_\gamma} + \frac{n_\nu-n_\gamma}{n}\ln(nw_2(n))}{\ln\frac{\sigma^2}{\sigma^2+c_\gamma} + \frac{n_\nu-n_\gamma}{n}\ln n} = 1.$$

(b) $\mathcal{M}_\nu$ is nested within $\mathcal{M}_\gamma$:
We obtain

$$\operatorname*{plim}_{n\to\infty} C_{\gamma\nu} = \operatorname*{plim}_{n\to\infty} \frac{(n-1)\ln\frac{(1-F_\nu^2)}{(1-F_\gamma^2)} + (n_\nu-n_\gamma)\ln(nw_2(n)) + 2\times\mathrm{Const}}{n\ln\frac{\mathrm{RSS}_\nu}{\mathrm{RSS}_\gamma} + (n_\nu-n_\gamma)\ln(nw_2(n))} = 1$$

due to $n_\gamma > n_\nu$ and $n\ln(\mathrm{RSS}_\nu/\mathrm{RSS}_\gamma) \xrightarrow{d} \chi^2_{n_\gamma-n_\nu}$.

### 3.3 Proof of Theorem 3

We now sketch the proof of Theorem 3. For the case that $\mathcal{M}_\nu$ is not nested within $\mathcal{M}_\gamma$, the proof is similar to that of Theorem 1. When $\mathcal{M}_\nu$ is nested within $\mathcal{M}_\gamma$, Lemma 5 shows the following relationship

$$\ln\left[\frac{b_\sigma + \mathbf{y}'(\mathbf{I}_n - \widetilde{\mathbf{K}}_\nu \boldsymbol{\Theta}_\nu^{-1} \widetilde{\mathbf{K}}_\nu')\mathbf{y}}{b_\sigma + \mathbf{y}'(\mathbf{I}_n - \widetilde{\mathbf{K}}_\gamma \boldsymbol{\Theta}_\gamma^{-1} \widetilde{\mathbf{K}}_\gamma')\mathbf{y}}\right] \le \ln\left[\frac{\mathbf{y}'(\mathbf{I}_n - \widetilde{\mathbf{K}}_\nu \boldsymbol{\Theta}_\nu^{-1} \widetilde{\mathbf{K}}_\nu')\mathbf{y}}{\mathbf{y}'(\mathbf{I}_n - \widetilde{\mathbf{K}}_\gamma \boldsymbol{\Theta}_\gamma^{-1} \widetilde{\mathbf{K}}_\gamma')\mathbf{y}}\right].$$

We thus have

$$\plim_{n\to\infty} \frac{a_\sigma + n}{2} \ln\left[\frac{b_\sigma + \mathbf{y}'(\mathbf{I}_n - \widetilde{\mathbf{K}}_\nu \boldsymbol{\Theta}_\nu^{-1} \widetilde{\mathbf{K}}_\nu')\mathbf{y}}{b_\sigma + \mathbf{y}'(\mathbf{I}_n - \widetilde{\mathbf{K}}_\gamma \boldsymbol{\Theta}_\gamma^{-1} \widetilde{\mathbf{K}}_\gamma')\mathbf{y}}\right] \le \plim_{n\to\infty} \frac{a_\sigma + n}{2} \ln\left[\frac{\mathbf{y}'(\mathbf{I}_n - \widetilde{\mathbf{K}}_\nu \boldsymbol{\Theta}_\nu^{-1} \widetilde{\mathbf{K}}_\nu')\mathbf{y}}{\mathbf{y}'(\mathbf{I}_n - \widetilde{\mathbf{K}}_\gamma \boldsymbol{\Theta}_\gamma^{-1} \widetilde{\mathbf{K}}_\gamma')\mathbf{y}}\right]$$

$$= \plim_{n\to\infty} \frac{a_\sigma + n}{2} \ln\left[\frac{\mathbf{y}'(\mathbf{I}_n - \widetilde{\mathbf{H}}_\nu)\mathbf{y}}{\mathbf{y}'(\mathbf{I}_n - \widetilde{\mathbf{H}}_\gamma)\mathbf{y}}\right] \in (0, \infty).$$

From this result the proof follows readily.

## 4 Conclusions

In this paper we have presented a frequentist analysis of a Bayesian model choice procedure for sparse regression. We have captured sparsity by a particular choice of prior distribution which we have referred to as a "Silverman $g$-prior." This prior emerges naturally from the RKHS perspective. It is similar in spirit to the Zellner $g$-prior, which has been widely used for Bayesian variable selection and Bayesian model selection due to its computational tractability in the evaluation of marginal likelihoods [6, 2]. Our analysis provides a theoretical foundation for the Silverman $g$-prior and suggests that it can play a similarly wide-ranging role in the development of fully Bayesian kernel methods.

## References

[1] C. Fernández, E. Ley, and M. F. J. Steel. Benchmark priors for Bayesian model averaging. *Journal of Econometrics*, 100:381–427, 2001.

[2] E. I. George and R. E. McCulloch. Approaches for Bayesian variable selection. *Statistica Sinica*, 7:339–374, 1997.

[3] R. E. Kass and A. E. Raftery. Bayes factors. *Journal of the American Statistical Association*, 90:773–795, 1995.

[4] F. Liang, R. Paulo, G. Molina, M. A. Clyde, and J. O. Berger. Mixtures of g-priors for Bayesian variable selection. *Journal of the American Statistical Association*, 103(481):410–423, 2008.

[5] B. W. Silverman. Some aspects of the spline smoothing approach to non-parametric regression curve fitting (with discussion). *Journal of the Royal Statistical Society, B*, 47(1):1–52, 1985.

[6] M. Smith and R. Kohn. Nonparametric regression using Bayesian variable selection. *Journal of Econometrics*, 75:317–344, 1996.

[7] G. Wahba. *Spline Models for Observational Data*. SIAM, Philadelphia, 1990.

[8] M. West. Bayesian factor regression models in the "large $p$, small $n$" paradigm. In J. M. Bernardo, M. J. Bayarri, J. .O Berger, A. P. Dawid, D. Heckerman, A. F. M. Smith, and M. West, editors, *Bayesian Statistics 7*, pages 723–732. Oxford University Press, 2003.

[9] A. Zellner. On assessing prior distributions and Bayesian regression analysis with $g-$prior distributions. In P. K. Goel and A. Zellner, editors, *Bayesian Inference and Decision Techniques: Essays in Honor of Bruno de Finetti*, pages 233–243. North-Holland, Amsterdam, 1986.
